# Evaluating probabilities under high-dimensional latent variable models

**Iain Murray and Ruslan Salakhutdinov**
Department of Computer Science
University of Toronto
Toronto, ON. M5S 3G4. Canada.
{murray,rsalakhu}@cs.toronto.edu

## Abstract

We present a simple new Monte Carlo algorithm for evaluating probabilities of observations in complex latent variable models, such as Deep Belief Networks. While the method is based on Markov chains, estimates based on short runs are formally unbiased. In expectation, the log probability of a test set will be underestimated, and this could form the basis of a probabilistic bound. The method is much cheaper than gold-standard annealing-based methods and only slightly more expensive than the cheapest Monte Carlo methods. We give examples of the new method substantially improving simple variational bounds at modest extra cost.

## 1   Introduction

Latent variable models capture underlying structure in data by explaining observations as part of a more complex, partially observed system. A large number of probabilistic latent variable models have been developed, most of which express a joint distribution $P(\mathbf{v}, \mathbf{h})$ over observed quantities $\mathbf{v}$ and their unobserved counterparts $\mathbf{h}$. Although it is by no means the only way to evaluate a model, a natural question to ask is "what probability $P(\mathbf{v})$ is assigned to a test observation?".

In some models the latent variables associated with a test input can be easily summed out: $P(\mathbf{v}) = \sum_{\mathbf{h}} P(\mathbf{v}, \mathbf{h})$. As an example, standard mixture models have a single discrete mixture component indicator for each data point; the joint probability $P(\mathbf{v}, \mathbf{h})$ can be explicitly evaluated for each setting of the latent variable.

More complex graphical models explain data through the combination of many latent variables. This provides richer representations, but provides greater computational challenges. In particular, marginalizing out many latent variables can require complex integrals or exponentially large sums. One popular latent variable model, the Restricted Boltzmann Machine (RBM), is unusual in that the posterior over hiddens $P(\mathbf{h}|\mathbf{v})$ is fully-factored, which allows efficient evaluation of $P(\mathbf{v})$ up to a constant. Almost all other latent variable models have posterior dependencies amongst latent variables, even if they are independent a priori.

Our current work is motivated by recent work on evaluating RBMs and their generalization to Deep Belief Networks (DBNs) [1]. For both types of models, a single constant was accurately approximated so that $P(\mathbf{v}, \mathbf{h})$ could be evaluated point-wise. For RBMs, the remaining sum over hidden variables was performed analytically. For DBNs, test probabilities were lower-bounded through a variational technique. Perhaps surprisingly, the bound was unable to reveal any significant improvement over RBMs in an experiment on MNIST digits. It was unclear whether this was due to looseness of the bound, or to there being no difference in performance.

A more accurate method for summing over latent variables would enable better and broader evaluation of DBNs. In section 2 we consider existing Monte Carlo methods. Some of them are certainly

more accurate, but prohibitively expensive for evaluating large test sets. We then develop a new cheap Monte Carlo procedure for evaluating latent variable models in section 3. Like the variational method used previously, our method is unlikely to spuriously over-state test-set performance. Our presentation is for general latent variable models, however for a running example, we use DBNs (see section 4 and [2]). The benefits of our new approach are demonstrated in section 5.

## 2 Probability of observations as a normalizing constant

The probability of a data vector, $P(\mathbf{v})$, is the normalizing constant relating the posterior over hidden variables to the joint distribution in Bayes rule, $P(\mathbf{h}|\mathbf{v}) = P(\mathbf{h}, \mathbf{v})/P(\mathbf{v})$. A large literature on computing normalizing constants exists in physics, statistics and computer science. In principle, there are many methods that could be applied to evaluating the probability assigned to data by a latent variable model. We review a subset of these methods, with notation and intuitions that will help motivate and explain our new algorithm.

In what follows, all auxiliary distributions $Q$ and transition operators $T$ are conditioned on the current test case $\mathbf{v}$, this is not shown in the notation to reduce clutter. Further, all of these methods assume that we can evaluate $P(\mathbf{h}, \mathbf{v})$. Graphical models with undirected connections will require the separate estimation of a single constant as in [1].

### 2.1 Importance sampling

Importance sampling can in principle find the normalizing constant of any distribution. The algorithm involves averaging a simple ratio under samples from some convenient tractable distribution over the hidden variables, $Q(\mathbf{h})$. Provided $Q(\mathbf{h}) \neq 0$ whenever $P(\mathbf{h}, \mathbf{v}) \neq 0$, we obtain:

$$P(\mathbf{v}) \ = \ \sum_{\mathbf{h}} \frac{P(\mathbf{h}, \mathbf{v})}{Q(\mathbf{h})} \, Q(\mathbf{h}) \ \approx \ \frac{1}{S} \sum_{s=1}^{S} \frac{P\big(\mathbf{h}^{(s)}, \mathbf{v}\big)}{Q\big(\mathbf{h}^{(s)}\big)}, \quad \mathbf{h}^{(s)} \sim Q\big(\mathbf{h}^{(s)}\big). \tag{1}$$

Importance sampling relies on the sampling distribution $Q(\mathbf{h})$ being similar to the target distribution $P(\mathbf{h}|\mathbf{v})$. Specifically, the variance of the estimator is an $\alpha$-divergence between the distributions [3]. Finding a tractable $Q(\mathbf{h})$ with small divergence is difficult in high-dimensional problems.

### 2.2 The Harmonic mean method

Using $Q(\mathbf{h}) = P(\mathbf{h}|\mathbf{v})$ in (1) gives an "estimator" that requires knowing $P(\mathbf{v})$. As an alternative, the harmonic mean method, also called the reciprocal method, gives an unbiased estimate of $1/P(\mathbf{v})$:

$$\frac{1}{P(\mathbf{v})} \ = \ \sum_{\mathbf{h}} \frac{P(\mathbf{h})}{P(\mathbf{v})} \ = \ \sum_{\mathbf{h}} \frac{P(\mathbf{h}|\mathbf{v})}{P(\mathbf{v}|\mathbf{h})} \ \approx \ \frac{1}{S} \sum_{s=1}^{S} \frac{1}{P\big(\mathbf{v}|\mathbf{h}^{(s)}\big)}, \quad \mathbf{h}^{(s)} \sim P\big(\mathbf{h}^{(s)}|\mathbf{v}\big). \tag{2}$$

In practice correlated samples from MCMC are used; then the estimator is asymptotically unbiased.

It was clear from the original paper and its discussion that the harmonic mean estimator can behave very poorly [4]. Samples in the tails of the posterior have large weights, which makes it easy to construct distributions where the estimator has infinite variance. A finite set of samples will rarely include any extremely large weights, so the estimator's empirical variance can be misleadingly low. In many problems, the estimate of $1/P(\mathbf{v})$ will be an underestimate with high probability. That is, the method will overestimate $P(\mathbf{v})$ and often give no indication that it has done so.

Sometimes the estimator will have manageable variance. Also, more expensive versions of the estimator exist with lower variance. However, it is still prone to overestimate test probabilities: If $1/\hat{P}_{\text{HME}}(\mathbf{v})$ is the Harmonic Mean Estimator in (2), Jensen's inequality gives $P(\mathbf{v}) = 1/\mathbb{E}\big[1/\hat{P}_{\text{HME}}(\mathbf{v})\big] \leq \mathbb{E}\big[\hat{P}_{\text{HME}}(\mathbf{v})\big]$. Similarly $\log P(\mathbf{v})$ will be overestimated in expectation. Hence the average of a large number of test log probabilities is highly likely to be an overestimate.

Despite these problems the estimator has received significant attention in statistics, and has been used for evaluating latent variable models in recent machine learning literature [5, 6]. This is understandable: all of the existing, more accurate methods are harder to implement and take considerably longer to run. In this paper we propose a method that is nearly as easy to use as the harmonic mean method, but with better properties.

## 2.3 Importance sampling based on Markov chains

Paradoxically, introducing auxiliary variables and making a distribution much higher-dimensional than it was before, can help find an approximating $Q$ distribution that closely matches the target distribution. As an example we give a partial review of Annealed Importance Sampling (AIS) [7], a special case of a larger family of Sequential Monte Carlo (SMC) methods (see, e.g., [8]). Some of this theory will be needed in the new method we present in section 3.

Annealing algorithms start with a sample from some tractable distribution $P_1$. Steps are taken with a series of operators $T_2, T_3, \ldots, T_S$, whose stationary distributions, $P_s$, are "cooled" towards the distribution of interest. The probability over the resulting sequence $H = \{\mathbf{h}^{(1)}, \mathbf{h}^{(2)}, \ldots \mathbf{h}^{(S)}\}$ is:

$$\mathcal{Q}_{\text{AIS}}(H) \ = \ P_1\big(\mathbf{h}^{(1)}\big) \prod_{s=2}^{S} T_s\big(\mathbf{h}^{(s)} \leftarrow \mathbf{h}^{(s-1)}\big). \tag{3}$$

To compute importance weights, we need to define a "target" distribution on the same state-space:

$$\mathcal{P}_{\text{AIS}}(H) \ = \ P\big(\mathbf{h}^{(S)}|\mathbf{v}\big) \prod_{s=2}^{S} \widetilde{T}_s\big(\mathbf{h}^{(s-1)} \leftarrow \mathbf{h}^{(s)}\big). \tag{4}$$

Because $\mathbf{h}^{(S)}$ has marginal $P(\mathbf{h}|\mathbf{v}) = P(\mathbf{h}, \mathbf{v})/P(\mathbf{v})$, $\mathcal{P}_{\text{AIS}}(H)$ has our target, $P(\mathbf{v})$, as its normalizing constant. The $\widetilde{T}$ operators are the *reverse operators*, of those used to define $\mathcal{Q}_{\text{AIS}}$.

For any transition operator $T$ that leaves a distribution $P(\mathbf{h}|\mathbf{v})$ stationary, there is a unique corresponding "reverse operator" $\widetilde{T}$, which is defined for any point $\mathbf{h}'$ in the support of $P$:

$$\widetilde{T}(\mathbf{h} \leftarrow \mathbf{h}') \ = \ \frac{T(\mathbf{h}' \leftarrow \mathbf{h})\, P(\mathbf{h}|\mathbf{v})}{\sum_{\mathbf{h}} T(\mathbf{h}' \leftarrow \mathbf{h})\, P(\mathbf{h}|\mathbf{v})} \ = \ \frac{T(\mathbf{h}' \leftarrow \mathbf{h})\, P(\mathbf{h}|\mathbf{v})}{P(\mathbf{h}'|\mathbf{v})}. \tag{5}$$

The sum in the denominator is known because $T$ leaves the posterior stationary. Operators that are their own reverse operator are said to satisfy "detailed balance" and are also known as "reversible". Many transition operators used in practice, such as Metropolis–Hastings, are reversible. Non-reversible operators are usually composed from a sequence of reversible operations, such as the component updates in a Gibbs sampler. The reverse of these (so-called) non-reversible operators is constructed from the same reversible base operations, but applied in reverse order.

The definitions above allow us to write:

$$\mathcal{Q}_{\text{AIS}}(H) = \mathcal{P}_{\text{AIS}}(H) \frac{\mathcal{Q}_{\text{AIS}}(H)}{\mathcal{P}_{\text{AIS}}(H)} = \mathcal{P}_{\text{AIS}}(H) \frac{P_1\big(\mathbf{h}^{(1)}\big)}{P\big(\mathbf{h}^{(S)}|\mathbf{v}\big)} \cdot \prod_{s=2}^{S} \frac{T_s\big(\mathbf{h}^{(s)} \leftarrow \mathbf{h}^{(s-1)}\big)}{\widetilde{T}_s\big(\mathbf{h}^{(s-1)} \leftarrow \mathbf{h}^{(s)}\big)}$$

$$= \mathcal{P}_{\text{AIS}}(H)\, P(\mathbf{v}) \left[ \frac{P_1\big(\mathbf{h}^{(1)}\big)}{P\big(\mathbf{h}^{(S)}, \mathbf{v}\big)} \cdot \prod_{s=2}^{S} \frac{P_s^*\big(\mathbf{h}^{(s)}\big)}{P_s^*\big(\mathbf{h}^{(s-1)}\big)} \right] \equiv \frac{\mathcal{P}_{\text{AIS}}(H)\, P(\mathbf{v})}{w(H)}. \tag{6}$$

We can usually evaluate the $P_s^*$, which are unnormalized versions of the stationary distributions of the Markov chain operators. Therefore the AIS importance weight $w(H) = 1/[\cdots]$ is tractable as long as we can evaluate $P(\mathbf{h}, \mathbf{v})$. The AIS importance weight provides an unbiased estimate:

$$\mathbb{E}_{\mathcal{Q}_{\text{AIS}}(H)}\Big[w(H)\Big] = P(\mathbf{v}) \sum_{H} \mathcal{P}_{\text{AIS}}(H) = P(\mathbf{v}). \tag{7}$$

As with standard importance sampling, the variance of the estimator depends on a divergence between $\mathcal{P}_{\text{AIS}}$ and $\mathcal{Q}_{\text{AIS}}$. This can be made small, at large computational expense, by using hundreds or thousands of steps $S$, allowing the neighboring intermediate distributions $P_s(\mathbf{h})$ to be close.

## 2.4 Chib-style estimators

Bayes rule implies that for any special hidden state $\mathbf{h}^*$, $P(\mathbf{v}) = P(\mathbf{h}^*, \mathbf{v})/P(\mathbf{h}^*|\mathbf{v})$. $\qquad\qquad$ (8)

This trivial identity suggests a family of estimators introduced by Chib [9]. First, we choose a particular hidden state $\mathbf{h}^*$, usually one with high posterior probability, and then estimate $P(\mathbf{h}^*|\mathbf{v})$.

We would like to obtain an estimator that is based on a sequence of states $H = \{\mathbf{h}^{(1)}, \mathbf{h}^{(2)}, \ldots, \mathbf{h}^{(S)}\}$ generated by a Markov chain that explores the posterior distribution $P(\mathbf{h}|\mathbf{v})$. The most naive estimate of $P(\mathbf{h}^*|\mathbf{v})$ is the fraction of states in $H$ that are equal to the special state $\sum_s \mathbb{I}(\mathbf{h}^{(s)} = \mathbf{h}^*)/S$.

Obviously this estimator is impractical as it equals zero with high probability when applied to high-dimensional problems. A "Rao–Blackwellized" version of this estimator, $\hat{p}(H)$, replaces the indicator function with the probability of transitioning from $\mathbf{h}^{(s)}$ to the special state under a Markov chain transition operator that leaves the posterior stationary. This can be derived directly from the operator's stationary condition:

$$P(\mathbf{h}^*|\mathbf{v}) = \sum_{\mathbf{h}} T(\mathbf{h}^* \leftarrow \mathbf{h})P(\mathbf{h}|\mathbf{v}) \approx \hat{p}(H) \equiv \frac{1}{S}\sum_{s=1}^{S} T(\mathbf{h}^* \leftarrow \mathbf{h}^{(s)}), \quad \{\mathbf{h}^{(s)}\} \sim \mathcal{P}(H), \quad (9)$$

where $\mathcal{P}(H)$ is the joint distribution arising from $S$ steps of a Markov chain. If the chain has stationary distribution $P(\mathbf{h}|\mathbf{v})$ and could be initialized at equilibrium so that

$$\mathcal{P}(H) = P\big(\mathbf{h}^{(1)}\big|\mathbf{v}\big)\prod_{s=2}^{S} T\big(\mathbf{h}^{(s)} \leftarrow \mathbf{h}^{(s-1)}\big), \qquad (10)$$

then $\hat{p}(H)$ would be an unbiased estimate of $P(\mathbf{h}^*|\mathbf{v})$. For ergodic chains the stationary distribution is achieved asymptotically and the estimator is consistent regardless of how it is initialized.

If $T$ is a Gibbs sampling transition operator, the only way of moving from $\mathbf{h}$ to $\mathbf{h}^*$ is to draw each element of $\mathbf{h}^*$ in turn. If updates are made in index order from 1 to $M$, the move has probability:

$$T(\mathbf{h}^* \leftarrow \mathbf{h}) = \prod_{j=1}^{M} P\big(h_j^* \,\big|\, \mathbf{h}_{1:(j-1)}^*,\, \mathbf{h}_{(j+1):M}\big). \qquad (11)$$

Equations (9, 11) have been used in schemes for monitoring the convergence of Gibbs samplers [10].

It is worth emphasizing that we have only outlined the simplest possible scheme inspired by Chib's general approach. For some Markov chains, there are technical problems with the above construction, which require an extension explained in the appendix. Moreover the approach above is *not* what Chib recommended. In fact, [11] explicitly favors a more elaborate procedure involving sampling from a sequence of distributions. This opens up the possibility of many sophisticated developments, e.g. [12, 13]. However, our focus in this work is on obtaining more useful results from simple cheap methods. There are also well-known problems with the Chib approach [14], to which we will return.

## 3 A new estimator for evaluating latent-variable models

We start with the simplest Chib-inspired estimator based on equations (8,9,11). Like many Markov chain Monte Carlo algorithms, (9) provides only (asymptotic) unbiasedness. For our purposes this is not sufficient. Jensen's inequality tells us

$$P(\mathbf{v}) = \frac{P(\mathbf{h}^*, \mathbf{v})}{P(\mathbf{h}^*|\mathbf{v})} = \frac{P(\mathbf{h}^*, \mathbf{v})}{\mathbb{E}[\hat{p}(H)]} \leq \mathbb{E}\left[\frac{P(\mathbf{h}^*, \mathbf{v})}{\hat{p}(H)}\right]. \qquad (12)$$

That is, we will overestimate the probability of a visible vector in expectation. Jensen's inequality also says that we will overestimate $\log P(\mathbf{v})$ in expectation.

Ideally we would like an accurate estimate of $\log P(\mathbf{v})$. However, if we must suffer some bias, then a lower bound that does not overstate performance will usually be preferred. An underestimate of $P(\mathbf{v})$ would result from overestimating $P(\mathbf{h}^*|\mathbf{v})$. The probability of the special state $\mathbf{h}^*$ will often be overestimated in practice if we initialize our Markov chain at $\mathbf{h}^*$. There are, however, simple counter-examples where this does not happen. Instead we describe a construction based on a sequence of Markov steps starting at $\mathbf{h}^*$ that does have the desired effect. We draw a state sequence from the following carefully designed distribution, using the algorithm in figure 1:

$$\mathcal{Q}(H) = \frac{1}{S}\sum_{s=1}^{S} \widetilde{T}\big(\mathbf{h}^{(s)} \leftarrow \mathbf{h}^*\big) \prod_{s'=s+1}^{S} T\big(\mathbf{h}^{(s')} \leftarrow \mathbf{h}^{(s'-1)}\big) \prod_{s'=1}^{s-1} \widetilde{T}\big(\mathbf{h}^{(s')} \leftarrow \mathbf{h}^{(s'+1)}\big). \qquad (13)$$

If the initial state were drawn from $P(\mathbf{h}|\mathbf{v})$ instead of $\widetilde{T}\big(\mathbf{h}^{(s)} \leftarrow \mathbf{h}^*\big)$, then the algorithm would give a sample from an equilibrium sequence with distribution $\mathcal{P}(H)$ defined in (10). This can be checked by repeated substitution of (5). This allows us to express $\mathcal{Q}$ in terms of $\mathcal{P}$, as we did for AIS:

$$\mathcal{Q}(H) = \frac{1}{S}\sum_{s=1}^{S} \frac{\widetilde{T}\big(\mathbf{h}^{(s)} \leftarrow \mathbf{h}^*\big)}{P\big(\mathbf{h}^{(s)}|\mathbf{v}\big)}\mathcal{P}(H) = \frac{1}{P(\mathbf{h}^*|\mathbf{v})}\left[\frac{1}{S}\sum_{s=1}^{S} T\big(\mathbf{h}^* \leftarrow \mathbf{h}^{(s)}\big)\right]\mathcal{P}(H). \qquad (14)$$

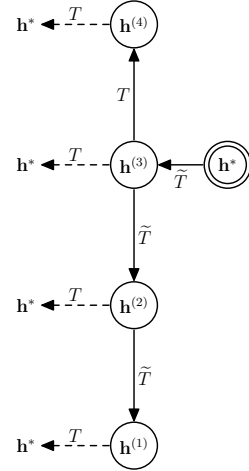

| Inputs: $\mathbf{v}$, observed test vector |
| --- |
| $\qquad$ $\mathbf{h}^*$, a (preferably high posterior probability) hidden state |
| $\qquad$ $S$, number of Markov chain steps |
| $\qquad$ $T$, Markov chain operator that leaves $P(\mathbf{h}|\mathbf{v})$ stationary |
| 1. Draw $s \sim \text{Uniform}(\{1, \dots S\})$ |
| 2. Draw $\mathbf{h}^{(s)} \sim \widetilde{T}\big(\mathbf{h}^{(s)} \leftarrow \mathbf{h}^*\big)$ |
| 3. **for** $s' = (s+1) : S$ |
| 4. $\qquad$ Draw $\mathbf{h}^{(s')} \sim T\big(\mathbf{h}^{(s')} \leftarrow \mathbf{h}^{(s'-1)}\big)$ |
| 5. **for** $s' = (s-1) : -1 : 1$ |
| 6. $\qquad$ Draw $\mathbf{h}^{(s')} \sim \widetilde{T}\big(\mathbf{h}^{(s')} \leftarrow \mathbf{h}^{(s'+1)}\big)$ |
| 7. $P(\mathbf{v}) \approx P(\mathbf{v},\mathbf{h}^*)\Big/ \dfrac{1}{S}\displaystyle\sum_{s'=1}^{S} T(\mathbf{h}^* \leftarrow \mathbf{h}^{(s')})$ |

Figure 1: Algorithm for the proposed method. The graphical model shows $\mathcal{Q}(H|s=3)$ for $S = 4$. At each generated state $T(\mathbf{h}^* \leftarrow \mathbf{h}^{(s')})$ is evaluated (step 7), roughly doubling the cost of sampling. The reverse operator, $\widetilde{T}$, was defined in section 2.3.

The quantity in square brackets is the estimator for $P(\mathbf{h}^*|\mathbf{v})$ given in (9). The expectation of the reciprocal of this quantity under draws from $\mathcal{Q}(H)$ is exactly the quantity needed to compute $P(\mathbf{v})$:

$$\mathbb{E}_{\mathcal{Q}(H)}\left[1 \Big/ \frac{1}{S}\sum_{s=1}^{S} T\big(\mathbf{h}^* \leftarrow \mathbf{h}^{(s)}\big)\right] = \frac{1}{P(\mathbf{h}^*|\mathbf{v})}\sum_{H}\mathcal{P}(H) = \frac{1}{P(\mathbf{h}^*|\mathbf{v})}. \qquad (15)$$

Although we are using the simple estimator from (9), by drawing $H$ from a carefully constructed Markov chain procedure, the estimator is now unbiased in $P(\mathbf{v})$. This is not an asymptotic result. As long as no division by zero has occurred in the above equations, the estimator is unbiased in $P(\mathbf{v})$ for finite runs of the Markov chain. Jensen's implies that $\log P(\mathbf{v})$ is underestimated in expectation.

Neal noted that Chibs method will return incorrect answers in cases where the Markov chain does not mix well amongst modes [14]. Our new proposed method will suffer from the same problem. Even if no transition probabilities are exactly zero, unbiasedness does not exclude being on a particular side of the correct answer with very high probability. Poor mixing may cause $P(\mathbf{h}^*|\mathbf{v})$ to be overestimated with high probability, which would result in an underestimate of $P(\mathbf{v})$, i.e., an overly conservative estimate of test performance.

The variance of the estimator is generally unknown, as it depends on the (generally unavailable) auto-covariance structure of the Markov chain. We can note one positive property: for the ideal Markov chain operator that mixes in one step, the estimator has zero variance and gives the correct answer immediately. Although this extreme will not actually occur, it does indicate that on easy problems, good answers can be returned more quickly than by AIS.

## 4 Deep Belief Networks

In this section we provide a brief overview of Deep Belief Networks (DBNs), recently introduced by [2]. DBNs are probabilistic generative models, that can contain many layers of hidden variables. Each layer captures strong high-order correlations between the activities of hidden features in the layer below. The top two layers of the DBN model form a Restricted Boltzmann Machine (RBM) which is an undirected graphical model, but the lower layers form a directed generative model. The original paper introduced a greedy, layer-by-layer unsupervised learning algorithm that consists of learning a stack of RBMs one layer at a time.

Consider a DBN model with two layers of hidden features. The model's joint distribution is:

$$P(\mathbf{v}, \mathbf{h}^1, \mathbf{h}^2) = P(\mathbf{v}|\mathbf{h}^1)\, P(\mathbf{h}^2, \mathbf{h}^1), \qquad (16)$$

where $P(\mathbf{v}|\mathbf{h}^1)$ represents a sigmoid belief network, and $P(\mathbf{h}^1, \mathbf{h}^2)$ is the joint distribution defined by the second layer RBM. By explicitly summing out $\mathbf{h}^2$, we can easily evaluate an unnormalized probability $P^*(\mathbf{v}, \mathbf{h}^1) = ZP(\mathbf{v}, \mathbf{h}^1)$. Using an approximating factorial posterior distribution $Q(\mathbf{h}|\mathbf{v})$,

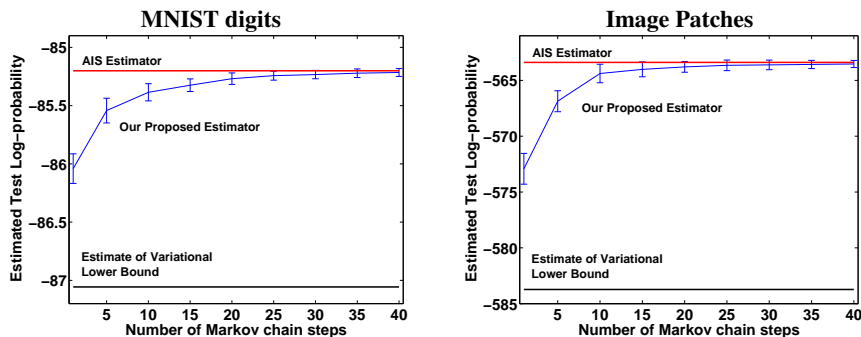

Figure 2: AIS, our proposed estimator and a variational method were used to sum over the hidden states for each of 50 randomly sampled test cases to estimate their average log probability. The three methods shared the same AIS estimate of a single global normalization constant $Z$.

obtained as a byproduct of the greedy learning procedure, and an AIS estimate of the model's partition function $Z$, [1] proposed obtaining an estimate of a variational lower bound:

$$\log P(\mathbf{v}) \geq \sum_{\mathbf{h}^1} Q(\mathbf{h}^1|\mathbf{v}) \log P^*(\mathbf{v}, \mathbf{h}^1) - \log Z + \mathcal{H}(Q(\mathbf{h}^1|\mathbf{v})). \tag{17}$$

The entropy term $\mathcal{H}(\cdot)$ can be computed analytically, since $Q$ is factorial, and the expectation term was estimated by a simple Monte Carlo approximation:

$$\sum_{\mathbf{h}^1} Q(\mathbf{h}^1|\mathbf{v}) \log P^*(\mathbf{v}, \mathbf{h}^1) \approx \frac{1}{S} \sum_{s=1..S} \log P^*(\mathbf{v}, \mathbf{h}^{1(s)}), \quad \text{where } \mathbf{h}^{1(s)} \sim Q(\mathbf{h}^1|\mathbf{v}). \tag{18}$$

Instead of the variational approach, we could also adopt AIS to estimate $\sum_{\mathbf{h}^1} P^*(\mathbf{v}, \mathbf{h}^1)$. This would be computationally very expensive, since we would need to run AIS for each test case.

In the next section we show that variational lower bounds can be quite loose. Running AIS on the entire test set, containing many thousands of test cases, is computationally too demanding. Our proposed estimator requires the same single AIS estimate of $Z$ as the variational method, so that we can evaluate $P(\mathbf{v}, \mathbf{h}^1)$. It then provides better estimates of $\log P(\mathbf{v})$ by approximately summing over $\mathbf{h}^1$ for each test case in a reasonable amount of computer time.

## 5 Experimental Results

We present experimental results on two datasets: the MNIST digits and a dataset of image patches, extracted from images of natural scenes taken from the collection of Van Hateren (http://hlab.phys.rug.nl/imlib/). The MNIST dataset contains 60,000 training and 10,000 test images of ten handwritten digits (0 to 9), with $28 \times 28$ pixels. The image dataset consisted of 130,000 training and 20,000 test $20 \times 20$ patches. The raw image intensities were preprocessed and whitened as described in [15]. Gibbs sampling was used as a Markov chain transition operator throughout. All log probabilities quoted use natural logarithms, giving values in nats.

### 5.1 MNIST digits

In our first experiment we used a deep belief network (DBN) taken from [1]. The network had two hidden layers with 500 and 2000 hidden units, and was greedily trained by learning a stack of two RBMs one layer at a time. Each RBM was trained using the Contrastive Divergence (CD) learning rule. The estimate of the lower bound on the average test log probability, using (17), was $-86.22$.

To estimate how loose the variational bound is, we randomly sampled 50 test cases, 5 of each class, and ran AIS for each test case to estimate the true test log probability. Computationally, this is equivalent to estimating 50 additional partition functions. Figure 2, left panel, shows the results. The estimate of the variational bound was $-87.05$ per test case, whereas the estimate of the true test log probability using AIS was $-85.20$. Our proposed estimator, averaged over 10 runs, provided an answer of $-85.22$. The special state $\mathbf{h}^*$ for each test example $\mathbf{v}$ was obtained by first sampling from the approximating distribution $Q(\mathbf{h}|\mathbf{v})$, and then performing deterministic hill-climbing in $\log p(\mathbf{v}, \mathbf{h})$ to get to a local mode.

AIS used a hand-tuned temperature schedule designed to equalize the variance of the intermediate log weights [7]. We needed 10,000 intermediate distributions to get stable results, which took about 3.6 days on a Pentium Xeon 3.00GHz machine, whereas for our proposed estimator we only used $S = 40$, which took about 50 minutes. For a more direct comparison we tried giving AIS 50 minutes, which allows 100 temperatures. This run gave an estimate of $-89.59$, which is lower than the lower bound and tells us nothing. Giving AIS ten times more time, 1000 temperatures, gave $-86.05$. This is higher than the lower bound, but still worse than our estimator at $S = 40$, or even $S = 5$.

Finally, using our proposed estimator, the average test log probability on the entire MNIST test data was $-84.55$. The difference of about 2 nats shows that the variational bound in [1] was rather tight, although a very small improvement of the DBN over the RBM is now revealed.

## 5.2 Image Patches

In our second experiment we trained a two-layer DBN model on the image patches of natural scenes. The first layer RBM had 2000 hidden units and 400 Gaussian visible units. The second layer represented a semi-restricted Boltzmann machine (SRBM) with 500 hidden and 2000 visible units. The SRBM contained visible-to-visible connections, and was trained using Contrastive Divergence together with mean-field. Details of training can be found in [15]. The overall DBN model can be viewed as a directed hierarchy of Markov random fields with hidden-to-hidden connections.

To estimate the model's partition function, we used AIS with 15,000 intermediate distributions and 100 annealing runs. The estimated lower bound on the average test log probability (see Eq. 17), using a factorial approximate posterior distribution $Q(\mathbf{h}^1|\mathbf{v})$, which we also get as a byproduct of the greedy learning algorithm, was $-583.73$. The estimate of the true test log probability, using our proposed estimator, was $-563.39$. In contrast to the model trained on MNIST, the difference of over 20 nats shows that, for model comparison purposes, the variational lower bound is quite loose.

For comparison, we also trained square ICA and a mixture of factor analyzers (MFA) using code from [16, 17]. Square ICA achieves a test log probability of $-551.14$, and MFA with 50 mixture components and a 30-dimensional latent space achieves $-502.30$, clearly outperforming DBNs.

## 6   Discussion

Our new Monte Carlo procedure is formally unbiased in estimating $P(\mathbf{v})$. In practice it is likely to underestimate the (log-)probability of a test set. Although the algorithm involves Markov chains, importance sampling underlies the estimator. Therefore the methods discussed in [18] could be used to bound the probability of accidentally over-estimating a test set probability.

In principle our procedure is a general technique for estimating normalizing constants. It would not always be appropriate however, as it would suffer the problems outlined in [14]. As an example our method will not succeed in estimating the global normalizing constant of an RBM.

For our method to work well, a state drawn from $\widetilde{T}(\mathbf{h}^{(s)} \leftarrow \mathbf{h}^*)$ should look like it could be part of an equilibrium sequence $H \sim \mathcal{P}(H)$. The details of the algorithm arose by developing existing Monte Carlo estimators, but the starting state $\mathbf{h}^{(s)}$ could be drawn from any arbitrary distribution:

$$\mathcal{Q}_{\text{var}}(H) \;=\; \frac{1}{S} \sum_{s=1}^{S} \frac{q(\mathbf{h}^{(s)})}{P(\mathbf{h}^{(s)}|\mathbf{v})} \mathcal{P}(H) \;=\; P(\mathbf{v}) \left[ \frac{1}{S} \sum_{s=1}^{S} \frac{q(\mathbf{h}^{(s)})}{P(\mathbf{h}^{(s)}, \mathbf{v})} \right] \mathcal{P}(H). \qquad (19)$$

As before the reciprocal of the quantity in square brackets would give an estimate of $P(\mathbf{v})$. If an approximation $q(\mathbf{h})$ is available that captures more mass than $\widetilde{T}(\mathbf{h} \leftarrow \mathbf{h}^*)$, this generalized estimator could perform better. We are hopeful that our method will be a natural next step in a variety of situations where improvements are sought over a deterministic approximation.

## Acknowledgments

This research was supported by NSERC and CFI. Iain Murray was supported by the government of Canada. We thank Geoffrey Hinton and Radford Neal for useful discussions, Simon Osindero for providing preprocessed image patches of natural scenes, and the reviewers for useful comments.

## References

[1] Ruslan Salakhutdinov and Iain Murray. On the quantitative analysis of Deep Belief Networks. In *Proceedings of the International Conference on Machine Learning*, volume 25, pages 872–879, 2008.

[2] Geoffrey E. Hinton, Simon Osindero, and Yee Whye Teh. A fast learning algorithm for deep belief nets. *Neural Computation*, 18(7):1527–1554, 2006.

[3] Tom Minka. Divergence measures and message passing. TR-2005-173, Microsoft Research, 2005.

[4] Michael A. Newton and Adrian E. Raftery. Approximate Bayesian inference with the weighted likelihood bootstrap. *Journal of the Royal Statistical Society, Series B (Methodological)*, 56(1):3–48, 1994.

[5] Thomas L. Griffiths, Mark Steyvers, David M. Blei, and Joshua B. Tenenbaum. Integrating topics and syntax. In *Advances in Neural Information Processing Systems (NIPS*17)*. MIT Press, 2005.

[6] Hanna M. Wallach. Topic modeling: beyond bag-of-words. In *Proceedings of the 23rd international conference on Machine learning*, pages 977–984. ACM Press New York, NY, USA, 2006.

[7] Radford M. Neal. Annealed importance sampling. *Statistics and Computing*, 11(2):125–139, 2001.

[8] Pierre Del Moral, Arnaud Doucet, and Ajay Jasra. Sequential Monte Carlo samplers. *Journal of the Royal Statistical Society B*, 68(3):1–26, 2006.

[9] Siddhartha Chib. Marginal likelihood from the Gibbs output. *Journal of the American Statistical Association*, 90(432):1313–1321, December 1995.

[10] Christian Ritter and Martin A. Tanner. Facilitating the Gibbs sampler: the Gibbs stopper and the griddy-Gibbs sampler. *Journal of the American Statistical Association*, 87(419):861–868, 1992.

[11] Siddhartha Chib and Ivan Jeliazkov. Marginal likelihood from the Metropolis–Hastings output. *Journal of the American Statistical Association*, 96(453), 2001.

[12] Antonietta Mira and Geoff Nicholls. Bridge estimation of the probability density at a point. *Statistica Sinica*, 14:603–612, 2004.

[13] Francesco Bartolucci, Luisa Scaccia, and Antonietta Mira. Efficient Bayes factor estimation from the reversible jump output. *Biometrika*, 93(1):41–52, 2006.

[14] Radford M. Neal. Erroneous results in "Marginal likelihood from the Gibbs output", 1999. Available from `http://www.cs.toronto.edu/~radford/chib-letter.html`.

[15] Simon Osindero and Geoffrey Hinton. Modeling image patches with a directed hierarchy of Markov random fields. In *Advances in Neural Information Processing Systems (NIPS*20)*. MIT Press, 2008.

[16] Aapo Hyvärinen. Fast and robust fixed-point algorithms for independent component analysis. *IEEE Transactions on Neural Networks*, 10(3):626–634, 1999.

[17] Zoubin Ghahramani and Geoffrey E. Hinton. The EM algorithm for mixtures of factor analyzers. Technical Report CRG-TR-96-1, University of Toronto, 1997.

[18] Vibhav Gogate, Bozhena Bidyuk, and Rina Dechter. Studies in lower bounding probability of evidence using the Markov inequality. In *23rd Conference on Uncertainty in Artificial Intelligence (UAI)*, 2007.

## A   Real-valued latents and Metropolis–Hastings

There are technical difficulties with the original Chib-style approach applied to Metropolis–Hastings and continuous latent variables. The continuous version of equation (9),

$$P(\mathbf{h}^*|\mathbf{v}) = \int T(\mathbf{h}^* \leftarrow \mathbf{h})P(\mathbf{h}|\mathbf{v})\,\mathrm{d}\mathbf{h} \approx \tfrac{1}{S}\sum_{s=1}^{S} T(\mathbf{h}^* \leftarrow \mathbf{h}^{(s)}), \quad \mathbf{h}^{(s)} \sim \mathcal{P}(H), \qquad (20)$$

doesn't work if $T$ is the Metropolis–Hastings operator. The Dirac-delta function at $\mathbf{h} = \mathbf{h}^*$ contains a significant part of the integral, which is ignored by samples from $P(\mathbf{h}|\mathbf{v})$ with probability one.

Following [11], the fix is to instead integrate over the generalized detailed balance relationship (5). Chib and Jeliazkov implicitly took out the $\mathbf{h}^* = \mathbf{h}$ point from all of their integrals. We do the same:

$$P(\mathbf{h}^*|\mathbf{v}) = \int \mathrm{d}\mathbf{h}\, \widetilde{T}(\mathbf{h}^* \leftarrow \mathbf{h})P(\mathbf{h}|\mathbf{v}) \Big/ \int \mathrm{d}\mathbf{h}\, T(\mathbf{h} \leftarrow \mathbf{h}^*). \qquad (21)$$

The numerator can be estimated as before. As both integrals omit $\mathbf{h} = \mathbf{h}^*$, the denominator is less than one when $T$ contains a delta function. For Metropolis–Hastings: $T(\mathbf{h} \leftarrow \mathbf{h}^*) = q(\mathbf{h}; \mathbf{h}^*)\min\big(1, a(\mathbf{h}; \mathbf{h}^*)\big)$, where $a(\mathbf{h}; \mathbf{h}^*)$ is an easy-to-compute acceptance ratio. Sampling from $q(\mathbf{h}; \mathbf{h}^*)$ and averaging $\min(1, a(\mathbf{h}; \mathbf{h}^*))$ provides an estimate of the denominator.

In our importance sampling approach there is no need to separately approximate an additional quantity. The algorithm in figure 1 still applies if the $T$'s are interpreted as probability density functions. If, due to a rejection, $\mathbf{h}^*$ is drawn in step 2. then the sum in step 7. will contain an infinite term giving a trivial underestimate $P(\mathbf{v}) = 0$. (Steps 3–6 need not be performed in this case.) On repeated runs, the average estimate is still unbiased, or an underestimate for chains that can't mix. Alternatively, the variational approach (19) could be applied together with Metropolis–Hastings sampling.